# PULSE-FIRING NEURAL CHIPS FOR HUNDREDS OF NEURONS

**Michael Brownlow**
**Lionel Tarassenko**
Dept. Eng. Science
Univ. of Oxford
Oxford OX1 3PJ

**Alan F. Murray**
Dept. Electrical Eng.
Univ. of Edinburgh
Mayfield Road
Edinburgh EH9 3JL

**Alister Hamilton**
**Il Song Han(1)**
**H. Martin Reekie**
Dept. Electrical Eng.
Univ. of Edinburgh

## ABSTRACT

We announce new CMOS synapse circuits using only three and four MOSFETs/synapse. Neural states are asynchronous pulse streams, upon which arithmetic is performed directly. Chips implementing over 100 fully programmable synapses are described and projections to networks of hundreds of neurons are made.

## 1 OVERVIEW OF PULSE FIRING NEURAL VLSI

The inspiration for the use of pulse firing in silicon neural networks is clearly the electrical/chemical pulse mechanism in "real" biological neurons. Asynchronous, digital voltage pulses are used to signal *states* $\{ S_i \}$ through *synapse weights* $\{ T_{ij} \}$ to emulate neural dynamics. Neurons fire voltage pulses of a *frequency* determined by their level of activity but of a constant magnitude (usually 5 Volts) [Murray,1989a]. As indicated in Fig. 1, synapses perform arithmetic directly on these asynchronous pulses, to increment or decrement the receiving neuron's activity. The activity of a receiving neuron $i$, $x_i$ is altered at a frequency controlled by the sending neuron $j$, with state $S_j$ by an amount determined by the synapse weight (here, $T_{ij}$).

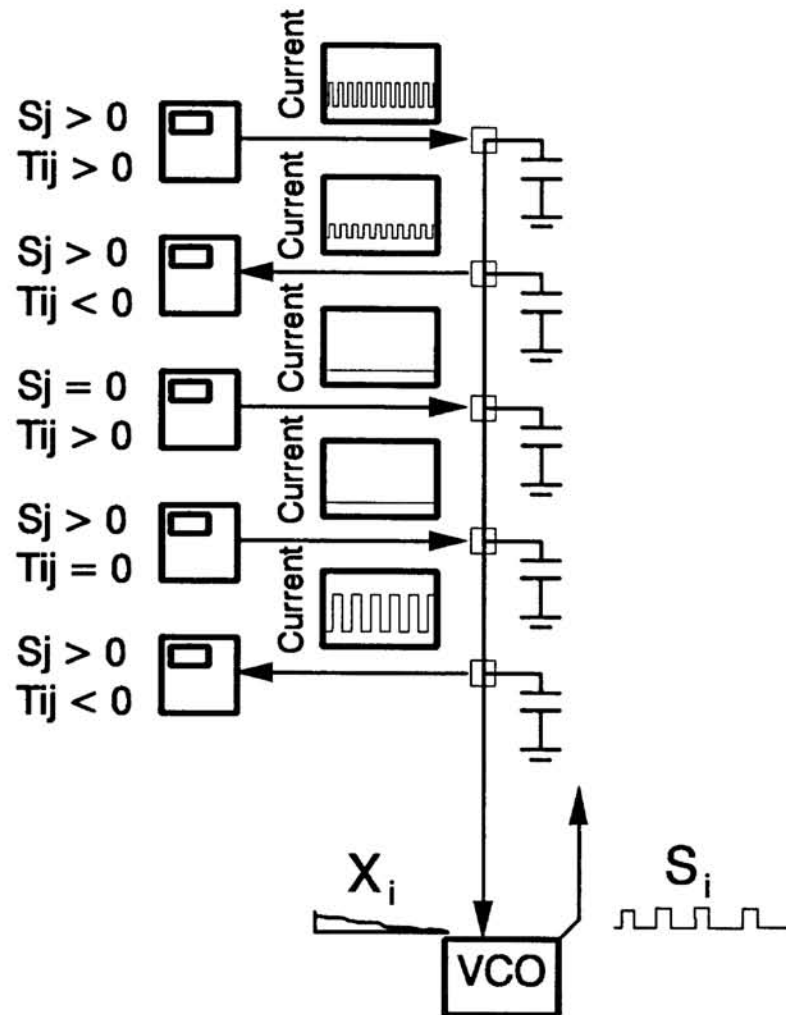

**Figure 1** : Pulse stream synapse functionality

A silicon neural network based on this technique is therefore an asynchronous, analog computational structure. It is a hybrid between analog and digital techniques in that the individual neural pulses are digital voltage spikes, with all the robustness to noise and ease of regeneration that this implies. These and other characteristics of pulse stream networks will be discussed in detail later in this paper. Pulse stream methods, developed in Edinburgh, have since been investigated by other groups - see for instance [El-Leithy,1988, Daniell,1989].

## 1.1. WHY PULSE STREAMS?

There are some advantages in the use of pulse streams, and pulse rate encoding, in implementing neural networks. It should be admitted here that the initial move towards pulse streams was motivated by the desire to implement pseudo-analog circuits on an essentially digital CMOS process. It was a decision based at the time on expediency rather than on great vision on our part, as we did not initially appreciate the full benefits of this form of pulse stream arithmetic [Murray,1987].

For example, the voltages on the terminals of a MOSFET, $V_{GS}$ and $V_{DS}$ could clearly be used to code a neural synapse weight and state respectively, doing away with the need for pulses. In the pulse stream form, however, we can arrange that only $V_{GS}$ is an "unknown". The device equations are therefore easily simplified, and furthermore the body effect is more predictable. In an equivalent continuous - time circuit, $V_{DS}$ will also be a variable, which codes information. Predicting the transistor's operating regime becomes more difficult, and the equation cannot be simplified. Aside of the transistor - level advantages, giving rise to extremely compact synapse circuits, there may be architectural advantages. There are certainly architectural **consequences**. Digital pulses are easier to regenerate, easier to pass between chips, and generally far more noise - insensitive than analog voltages, all of which are significant advantages in the VLSI context. Furthermore, the relationship to the biological exemplar should not be ignored. It is at least interesting - whether it is significant remains to be seen.

## 2 FULLY ANALOG PULSE STREAM SYNAPSES

Our early pulse stream chips proved the viability of the pulse stream technique [Murray,1988a]. However, the area occupied by the digital weight storage memory was unacceptably large. Furthermore, the use of pseudo-clocks in an analog circuit was both aesthetically unsatisfactory and detrimental to smooth dynamical behaviour, and using separate signal paths for excitation and inhibition was both clumsy and inefficient. Accordingly, we have developed a family of fully programmable, fully analog synapses using dynamic weight storage, and operating on individual pulses to perform arithmetic. We have already reported time-modulation synapses based on this technique, and a later section of this paper will present the associated chips [Murray,1988b, Murray,1989b].

### 2.1. TRANSCONDUCTANCE MULTIPLIER SYNAPSES

The equation of interest is that for the drain-source current, $I_{DS}$, for a MOSFET in the *linear* or *triode* region:-

$$I_{DS} = \frac{\mu C_{ox} W}{L} \left[ ( V_{GS} - V_T ) V_{DS} - \frac{V_{DS}^2}{2} \right] \tag{1}$$

Here, $C_{ox}$ is the oxide capacitance/area, $\mu$ the carrier mobility, W the transistor gate width, L the transistor gate length, and $V_{GS}$, $V_T$, $V_{DS}$ the transistor gate-source, threshold and drain-source voltages respectively.

This expression for $I_{DS}$ contains a useful product term:- $\frac{\mu C_{ox} W}{L} \times V_{GS} \times V_{DS}$.
However, it also contains two other terms in $V_{DS} \times V_T$ and $V_{DS}^2$.

One approach might be to ignore this imperfection in the multiplication, in the hope that the neural parallelism renders it irrelevant. We have chosen, rather, **to remove** the unwanted terms via a second MOSFET, as shown in Fig. 2.

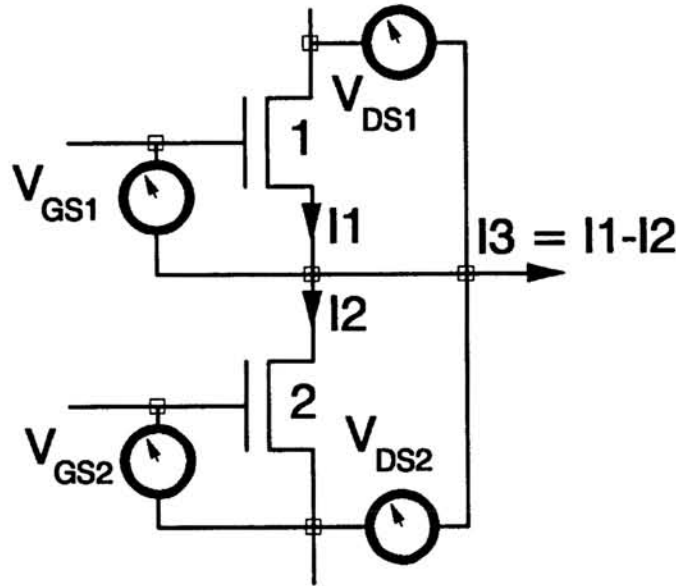

**Figure 2** : Use of a second MOSFET to remove nonlinearities
(a transconductance multiplier).

The output current $I_3$ is now given by:-

$$I_3 = \mu C_{ox} \left[ \frac{W_1}{L_1} ( V_{GS1} - V_T ) V_{DS1} - \frac{W_1}{L_1} \frac{V_{DS1}^2}{2} \right. \tag{2}$$

$$\left. - \frac{W_2}{L_2} ( V_{GS2} - V_T ) V_{DS2} + \frac{W_2}{L_2} \frac{V_{DS2}^2}{2} \right]$$

The secret now is to select $W_1, L_1, W_2, L_2, V_{GS1}, V_{GS2}, V_{DS1}$ and $V_{DS2}$ to cancel all terms except

$$\mu C_{ox} \frac{W_1}{L_1} V_{GS1} \times V_{DS1} \tag{3}$$

This is a fairly well-known circuit, and constitutes a **Transconductance Multiplier**. It was reported initially for use in signal processing chips such as filters [Denyer,1981, Han,1984]. It would be feasible to use it directly in a continuous time network, with analog voltages representing the $\{ S_i \}$. We choose to use it within a pulse-stream environment, to minimise the uncertainty in determining the operating regime, and terminal voltages, of the MOSFETs, as described above.

Fig. 3 shows two related pulse stream synapse based on this technique. The presynaptic neural state $S_j$ is represented by a stream of 0-5V digital, asynchronous voltage pulses $V_j$. These are used to switch a current sink and source in and out of the synapse, either pouring current **to** a fixed voltage node (excitation of the postsynaptic neuron), or removing it (inhibition). The magnitude and direction of the resultant current pulses are determined by the synapse weight, currently stored as a dynamic, analog voltage $T_{ij}$.

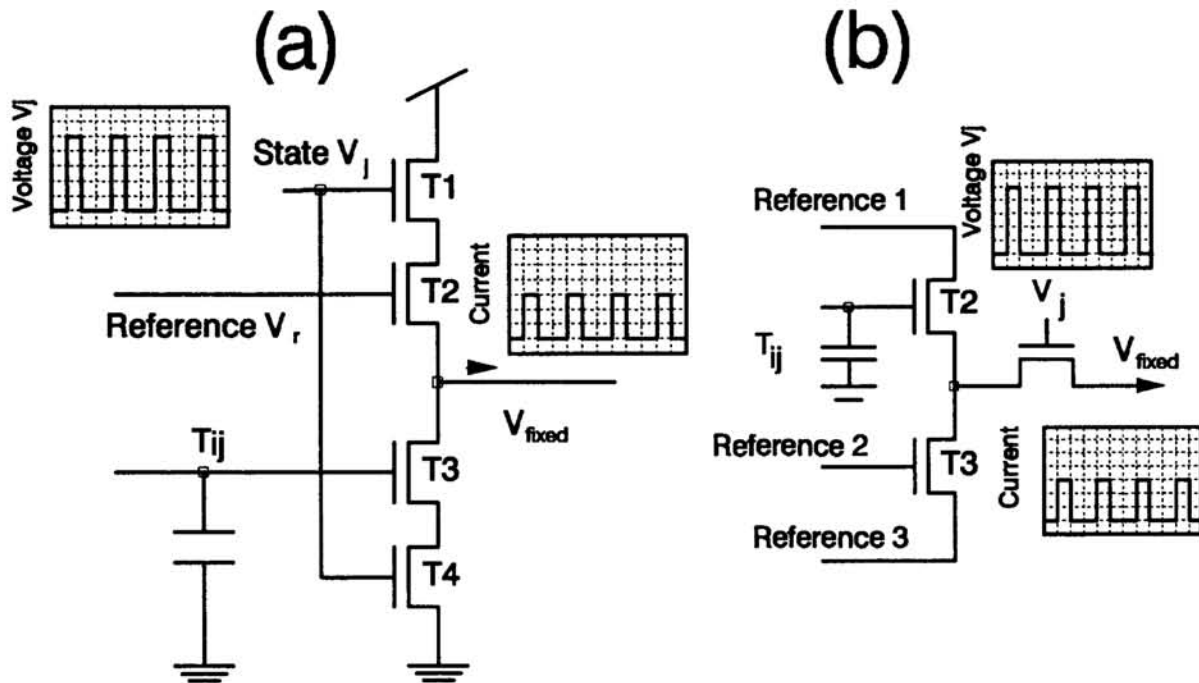

**Figure 3** : Use of a transconductance multiplier to
form fully programmable pulse-stream synapses.

The fixed voltage $V_{fixed}$ and the summation of the current pulses to give an
activity $x_i = \sum T_{ij} S_j$ are both provided by an Operational Amplifier
integrator circuit, whose saturation characteristics incidentally apply a
sigmoid nonlinearity. The transistors T1 and T4 act as power supply
"on/off" switches in Fig. 3a, and in Fig 3b are replaced by a single
transistor, in the output "leg" of the synapse, Transistors T2 and T3 form the
transconductance multiplier. One of the transistors has the synapse voltage
$T_{ij}$ on its gate, the other a reference voltage, whose value determines the
crossover point between excitation and inhibition. The gate-source voltages
on T2 and T3 need to be substantially greater than the drain-source
voltages, to maintain linear operation. This is not a difficult constraint to
satisfy.

The attractions of these cells are that all the transistors are n-type, removing
the need for area-hungry isolation well structures, and In Fig. 3a, the
vertical line of drain-source connections is topologically attractive,
producing very compact layout, while Fig. 3b has fewer devices. It is not
yet clear which will prove optimal.

## 2.2. ASYNCHRONOUS "SWITCHED CAPACITOR" SYNAPSE

Fig. 4 shows a further variant, in the form of a "switched capacitor" pulse
stream synapse. Here the synapse voltage $T_{ij}$ is electrically buffered to
switched capacitor structure, clocked *by the presynaptic neural pulse
waveforms*. Packets of charge are therefore "metered out" to the current
integrator whose magnitude is controlled by $T_{ij}$ (positive or negative), and

whose frequency by the presynaptic pulse rate. The overall principle is therefore the same as that described for the transconductance multiplier synapses, although the circuit level details are different.

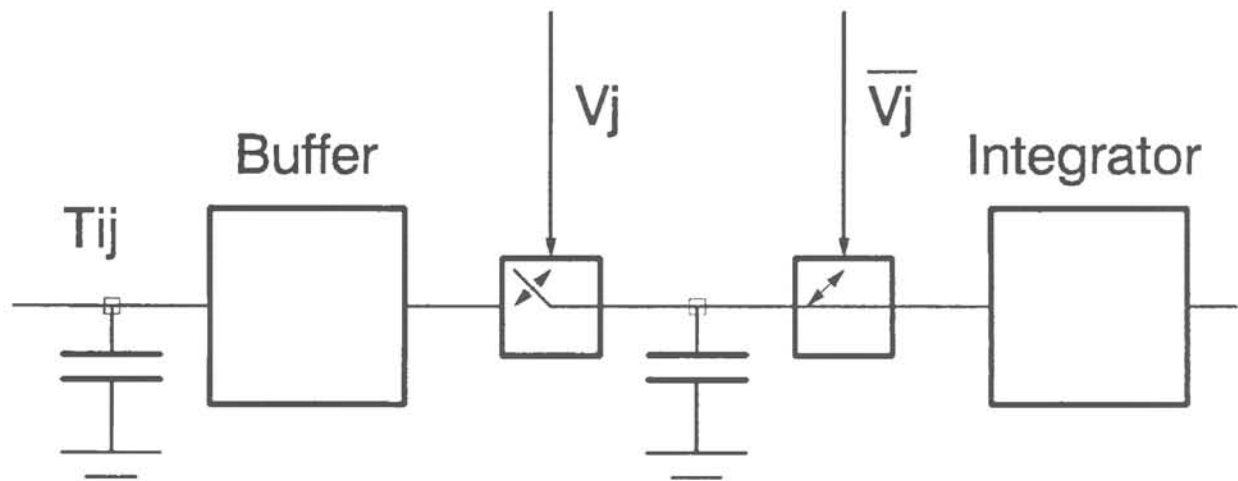

**Figure 4** : Asynchronous, "switched capacitor" pulse stream synapse.

Conventional synchronous switched capacitor techniques have been used in neural integration [Tsividis,1987], but nowhere as directly as in this example.

## 2.3. CHIP DETAILS AND RESULTS
Both the time-modulation and switched capacitor synapses have been tested fully in silicon, and Fig. 5 shows a section of the time-modulation test chip. This synapse currently occupies $174 \times 73 \mu$m.

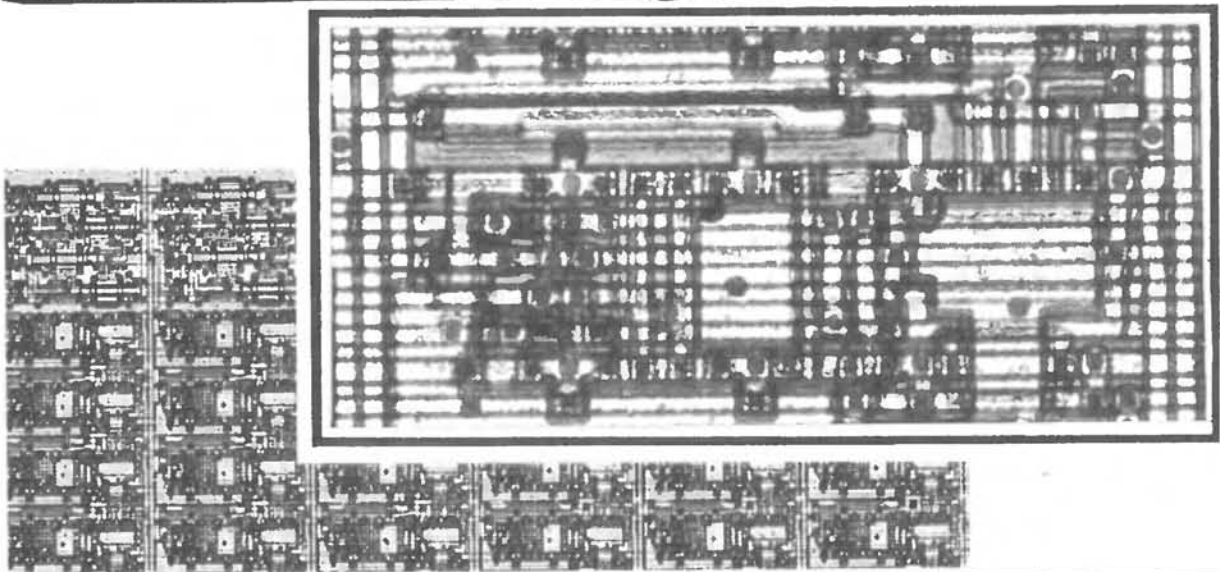

**Figure 5** : Section, and single synapse, from time-modulation chip.

Three distinct pulse-stream synapse types have been presented, with different operating schemes and characteristics. None has yet been used to configure a large network, but this is now being done. Current estimates for the number of synapses implementable using the two techniques described above are as shown in Table 1, using an 8mmx8mm die as an example.

The lack of direct scaling between transistor count and synapse count (e.g. why does the factor 4/11 not manifest itself as a much larger increase in synapse count) can be explained. The raw number of transistors is not the only factor in determining circuit area. Routing of power supplies, synapse weight address lines, as well as storage capacitor size all take their toll, and are common to both of the above synapse circuits. Furthermore, in analog circuitry, transistors are almost certainly larger than minimum geometry, and generally significantly larger, to minimise noise problems. This all gives rise to a larger area than might be expected from simple arguments. Clearly, however, we are in position to implement serious sized networks, firstly with the time-modulation synapse, which is fully tested in silicon, and later with the transconductance type, which is still under detailed design and layout.

Table 1 : Estimated synapse count on 8mm die

| SYNAPSE | NO. OF TRANSISTORS | ESTIMATED NETWORK SIZE |
|---|---|---|
| Time modulation | 11 | ≃6400 synapses |
| Transconductance | 4 | ≃15000 synapses |
| Switched Capacitor | 4 | ≃14000 synapses |

In addition, we are developing new oscillator forms, techniques to counteract leakage from dynamic nodes, novel inter-chip signalling strategies specifically for pulse-stream systems, and non-volatile ($\alpha$-Si) pulse stream synapses. These are to be used for applications in text-speech synthesis, pattern analysis and robotics. Details will be published as the work progresses.

**Acknowledgements**

The authors are grateful to the UK Science and Engineering Research Council, and the European Community (ESPRIT BRA) for its support of this work. Dr. Han is grateful to the Korean Telecommunications Authority, from whence he is on secondment in Edinburgh, and KOSEF(Korea) for partial financial support.

## Footnotes

[1] On secondment from the Korean Telecommunications Authority

## References

Daniell,1989.
    P. M. Daniell, W. A. J. Waller, and D. A. Bisset, "An Implementation of Fully Analogue Sum-of-Product Neural Models," *Proc. IEE Conf. on Artificial Neural Networks*, pp. 52-56, ,1989.

Denyer,1981.
    P. B. Denyer and J. Mavor, "MOST Transconductance Multipliers for Array Applications," *IEE Proc. Pt. 1*, vol. 128, no. 3, pp. 81-86, June ,1981.

El-Leithy,1988.
    N. El-Leithy, M. Zaghloul, and R. W. Newcomb, "Implementation of Pulse-Coded Neural Networks," *Proc. 27th Conf. on Decision and Control*, pp. 334-336, ,1988.

Han,1984.
    Il S. Han and Song B. Park, "Voltage-Controlled Linear Resistors by MOS Transistors and their Application to Active RC Filter MOS Integration," *Proc. IEEE*, pp. 1655-1657, Nov., ,1984.

Murray,1987.
    A. F. Murray and A. V. W. Smith, "Asynchronous Arithmetic for VLSI Neural Systems," *Electronics Letters*, vol. 23, no. 12, pp. 642-3, June, ,1987.

Murray,1988a.
    A. F. Murray and A. V. W. Smith, "Asynchronous VLSI Neural Networks using Pulse Stream Arithmetic," *IEEE Journal of Solid-State Circuits and Systems*, vol. 23, no. 3, pp. 688-697, June, ,1988.

Murray,1988b.
    A. F. Murray, L. Tarassenko, and A. Hamilton, "Programmable Analogue Pulse-Firing Neural Networks," *Neural Information Processing Systems Conference*, pp. 671-677, Morgan Kaufmann, ,1988.

Murray,1989a.
    A. F. Murray, "Pulse Arithmetic in VLSI Neural Networks," *IEEE MICRO*, vol. 9, no. 6, pp. 64-74, ,1989.

Murray,1989b.
    A. F. Murray, A. Hamilton, H. M. Reekie, and L. Tarassenko, "Pulse - Stream Arithmetic in Programmable Neural Networks," *Int. Symposium on Circuits and Systems, Portland, Oregon*, pp. 1210-1212, IEEE, ,1989.

Tsividis,1987.
    Y. P. Tsividis and D. Anastassiou, "Switched - Capacitor Neural Networks," *Electronics Letters*, vol. 23, no. 18, pp. 958 - 959, August, ,1987.
